# Learning the 2-D Topology of Images

**Nicolas Le Roux**
University of Montreal
nicolas.le.roux@umontreal.ca

**Yoshua Bengio**
University of Montreal
yoshua.bengio@umontreal.ca

**Pascal Lamblin**
University of Montreal
lamblinp@umontreal.ca

**Marc Joliveau**
École Centrale Paris
marc.joliveau@ecp.fr

**Balázs Kégl**
LAL/LRI, University of Paris-Sud, CNRS
91898 Orsay, France
kegl@lal.in2p3.fr

## Abstract

We study the following question: is the two-dimensional structure of images a very strong prior or is it something that can be learned with a few examples of natural images? If someone gave us a learning task involving images for which the two-dimensional topology of pixels was not known, could we discover it automatically and exploit it? For example suppose that the pixels had been permuted in a fixed but unknown way, could we recover the relative two-dimensional location of pixels on images? The surprising result presented here is that not only the answer is yes, but that about as few as a thousand images are enough to approximately recover the relative locations of about a thousand pixels. This is achieved using a manifold learning algorithm applied to pixels associated with a measure of distributional similarity between pixel intensities. We compare different topology-extraction approaches and show how having the two-dimensional topology can be exploited.

## 1 Introduction

Machine learning has been applied to a number of tasks involving an input domain with a special topology: one-dimensional for sequences, two-dimensional for images, three-dimensional for videos and for 3-D capture. Some learning algorithms are generic, e.g., working on arbitrary unstructured vectors in $\mathbb{R}^d$, such as ordinary SVMs, decision trees, neural networks, and boosting applied to generic learning algorithms. On the other hand, other learning algorithms successfully exploit the specific topology of their input, e.g., SIFT-based machine vision [10], convolutional neural networks [6, 7], time-delay neural networks [5, 16].

It has been conjectured [8, 2] that the two-dimensional structure of natural images is a very strong prior that would require a huge number of bits to specify, if starting from the completely uniform prior over all possible permutations.

The question studied here is the following: is the two-dimensional structure of natural images a very strong prior or is it something that can be learned with a few examples? If a small number of examples is enough to discover that structure, then the conjecture in [8] about the image topology was probably incorrect. To answer that question we consider a hypothetical learning task involving images whose pixels have been permuted in a fixed but unknown way. Could we recover the

two-dimensional relations between pixels automatically? Could we exploit it to obtain better generalization? A related study performed in the context of ICA can be found in [1].

The basic idea of the paper is that the two-dimensional topology of pixels can be recovered by looking for a two-dimensional manifold embedding pixels (each pixel is a point in that space), such that nearby pixels have similar distributions of intensity (and possibly color) values.

We explore a number of manifold techniques with this goal in mind, and explain how we have adapted these techniques in order to obtain the positive and surprising result: the two-dimensional structure of pixels can be recovered from a rather small number of training images. On images we find that the first 2 dimensions are dominant, meaning that even the knowledge that 2 dimensions are most appropriate could probably be inferred from the data.

## 2 Manifold Learning Techniques Used

In this paper we have explored the question raised in the introduction for the particular case of images, i.e., with 2-dimensional structures, and our experiments have been performed with images of size $27 \times 27$ to $30 \times 30$, i.e., with about a thousand pixels. It means that we have to look for the embedding of about a thousand points (the pixels) on a two-dimensional manifold. Metric Multi-Dimensional Scaling MDS is a linear embedding technique (analogous to PCA but starting from distances and yielding coordinates on the principal directions, of maximum variance). Nonparametric techniques such as Isomap [13], Local Linear Embedding (LLE) [12], or Semidefinite Embedding (SDE, also known as MVU for Maximum Variance Unfolding) [17] have computation time that scale polynomially in the number of examples $n$. With $n$ around a thousand, all of these are feasible, and we experimented with MDS, Isomap, LLE, and MVU.

Since we found Isomap to work best to recover the pixel topology even on small sets of images, we review the basic elements of Isomap. It applies the metric multidimensional scaling (MDS) algorithm to *geodesic distances in the neighborhood graph*. The neighborhood graph is obtained by connecting the $k$ nearest neighbors of each point. Each arc of the graph is associated with a distance (the user-provided distance between points), and is used to compute an approximation of the geodesic distance on the manifold with the length of the shortest path between two points. The metric MDS algorithm then transforms these distances into $d$-dimensional coordinates as follows. It first computes the dot-product (or Gram) $n \times n$ matrix $M$ using the "double-centering" formula, yielding entries $M_{ij} = -\frac{1}{2}(D_{ij}^2 - \frac{1}{n}\sum_i D_{ij}^2 - \frac{1}{n}\sum_j D_{ij}^2 + \frac{1}{n^2}\sum_{i,j} D_{ij}^2)$. The $d$ principal eigenvectors $v_k$ and eigenvalues $\lambda_k$ ($k = 1, \ldots, d$) of $M$ are then computed. This yields the coordinates: $x_{ik} = v_{ki}\sqrt{\lambda_k}$ is the $k$-th embedding coordinate of point $i$.

## 3 Topology-Discovery Algorithms

In order to apply a manifold learning algorithm, we must generally have a notion of similarity or distance between the points to embed. Here each point corresponds to a pixel, and the data we have about the pixels provide an empirical distribution of intensities for all pixels. Therefore we want to compare two estimate the statistical dependency between two pixels, in order to determine if they should be "neighbors" on the manifold. A simple and natural dependency statistic is the correlation between pixel intensities, and it works very well.

The empirical correlation $\rho_{ij}$ between the intensity of pixel $i$ and pixel $j$ is in the interval $[-1, 1]$. However, two pixels highly anti-correlated are much more likely to be close than pixels not correlated (think of edges in an image). We should thus consider the absolute value of the correlations. If we assume them to be the value of a Gaussian kernel

$$|\rho_{ij}| = K(x_i, x_j) = e^{-\frac{1}{2}\|x_i - x_j\|^2} ,$$

then by defining $D_{ij} = \|x_i - x_j\|$ and solving the above for $D_{ij}$ we obtain a "distance" formula that can be used with the manifold learning algorithms:

$$D_{ij} = \sqrt{-\log|\rho_{ij}|} . \tag{1}$$

Note that scaling the distances in the Gaussian kernel by a variance parameter would only scale the resulting embedding, so it is unnecessary.

Many other measures of distance would probably work as well. However, we found the absolute correlation to be simple and easy to understand while yielding nice embeddings.

### 3.1 Dealing With Low-Variance Pixels

A difficulty we observed in experimenting with different manifold learning algorithms on data sets such as MNIST is the influence of low-variance pixels. On MNIST digit images the border pixels may have 0 or very small variance. This makes them all want to be close to each other, which tends to fold the manifold on itself.

To handle this problem we have simply ignored pixels with very low variance. When these represent a fixed background (as in MNIST images), this strategy works fine. In the experiments with MNIST we removed pixels with standard deviation less than 15% of the maximum standard deviation (maximum over all pixels). On the NORB dataset, which has varied backgrounds, this step does not remove *any* of the pixels (so it is unnecessary).

## 4 Converting Back to a Grid Image

Once we have obtained an embedding for the pixels, the next thing we would like to do is to transform the data vectors back into images. For this purpose we have performed the following two steps:

1. Choosing horizontal and vertical axes (since the coordinates on the manifold can be arbitrarily rotated), and rotating the embedding coordinates accordingly, and

2. Transforming the input vector of intensity values (along with the pixel coordinates) into an ordinary discrete image on a grid. This should be done so that the resulting intensity at position $(i, j)$ is close to the intensity values associated with input pixels whose embedding coordinates are $(i, j)$.

Such a mapping of pixels to a grid has already been done in [4], where a grid topology is defined by the connections in a graphical model, which is then trained by maximizing the approximate likelihood. However, they are not starting from a continuous embedding, but from the original data.

Let $p_k$ $(k = 1 \ldots N)$ be the embedding coordinates found by the dimensionality reduction algorithm for the $k$-th input variable. We select the horizontal axis as the direction of smaller spread, the vertical axis being in the orthogonal direction, and perform the appropriate rotation.

Once we have a coordinate system that assigns a 2-dimensional position $p_k$ to the $k$-th input pixel, placed at irregular locations inside a rectangular grid, we can map the input intensities $x_k$ into intensities $M_{i,j}$, so as to obtain a regular image that can be processed by standard image-processing and machine vision learning algorithms. The output image pixel intensity $M_{i,j}$ at coordinates $(i, j)$ is obtained through a convex average

$$M_{i,j} = \sum_k w_{i,j,k} x_k \tag{2}$$

where the weights are non-negative and sum to one, and are chosen as follows.

$$w_{i,j,k} = \frac{v_{i,j,k}}{\sum_k v_{i,j,k}}$$

with an exponential of the $L_1$ distance to give less weight to farther points:

$$v_{i,j,k} = \exp\left(\gamma \|(i, j) - p_k\|_1\right) \mathbb{1}_{N(i,j,k)} \tag{3}$$

where $N(i, j, k)$ is true if $\|(i, j) - p_k\|_1 < 2$ (or inferior to a larger radius to make sure that at least one input pixel $k$ is associated with output grid position $(i, j)$). We used $\gamma = 3$ in the experiments, after trying only $1, 3$ and $10$. Large values of $\gamma$ correspond to using only the nearest neighbor of $(i, j)$ among the $p_k$s. Smaller values smooth the intensities and make the output look better if the embedding is not perfect. Too small values result in a loss of effective resolution.

**Algorithm 1** Pseudo-code of the topology-learning learning that recovers the 2-D structure of inputs provided in an arbitrary but fixed order.

---

**Input:** $X$       {Raw input $n \times N$ data matrix, one row per example, with elements in fixed but arbitrary order}

**Input:** $\delta = 0.15$ (default value){Minimum relative standard deviation threshold, to remove too low-variance pixels}

**Input:** $k = 4$ (default value){Number of neighbors used to build Isomap neighborhood graph}

**Input:** $L = \sqrt{N}, W = \sqrt{N}$ (default values) {Dimensions (length $L$, width $W$ of output image)}

**Input:** $\gamma = 3$ (default value) {Smoothing coefficient to recover images}

**Output:** $p$       {$N \times 2$ matrix of embedding coordinates (one per row) for each input variable}

**Output:** $w$       {Convolution weights to recover an image from a raw input vector}

  $n = $ number of examples (rows of $X$)

  **for all** column $X_{.i}$ **do**

    $\mu_i \leftarrow \frac{1}{n} \sum_t X_{ti}$ {Compute means}

    $\sigma_i^2 \leftarrow \frac{1}{n} \sum_t (X_{ti} - \mu_i)^2$ {Compute variances}

  **end for**

  Remove columns of $X$ for which $\frac{\sigma_i}{\max_j \sigma_j} < \delta$

  **for all** column $X_{.i}$ **do**

    **for all** column $X_{.j}$ **do**

      empirical correlation $\rho_{ij} = \frac{(X_{.i} - \mu_i)'(X_{.j} - \mu_j)}{\sigma_i \sigma_j}$ {Compute all pair-wise empirical correlations}

      pseudo-distances $D_{ij} = \sqrt{-\log|\rho_{ij}|}$

    **end for**

  **end for**

  {Compute the 2-D embeddings $(p_{k1}, p_{k2})$ of each input variable $k$ through Isomap}

  $p = \text{Isomap}(D, k, 2)$

  {Rotate the coordinates $p$ to try to align them to a vertical-horizontal grid (see text)}

  {Invert the axes if $L < W$}

  {Compute the convolution weights that will map raw values to output image pixel intensities}

  **for all** grid position $(i, j)$ in output image ($i$ in $1 \ldots L$, $j$ in $1 \ldots W$) **do**

    $r = 1$

    **repeat**

      neighbors $\leftarrow \{k : ||p_k - (i, j)||_1 < r\}$

      $r \leftarrow r + 1$

    **until** neighbors not empty

    **for all** $k$ in neighbors **do**

      $v_k \leftarrow e^{\gamma ||p_k - (i,j)||_1}$

    **end for**

    $w_{i,j,.} \leftarrow 0$

    **for all** $k$ in neighbors **do**

      $w_{i,j,k} = \frac{v_{i,j,k}}{\sum_k v_{i,j,k}}$ {Compute convolution weights}

    **end for**

  **end for**

---

**Algorithm 2** Convolve a raw input vector into a regular grid image, using the already discovered embedding for each input variable.

---

**Input:** $x$       {Raw input $N$-vector (in same format as a row of $X$ above)}

**Input:** $p$       {$N \times 2$ matrix of embedding coordinates (one per row) for each input variable}

**Input:** $w$       {Convolution weights to recover an image from a raw input vector}

**Output:** $Y$       {$L \times W$ output image}

  **for all** grid position $(i, j)$ in output image ($i$ in $1 \ldots L$, $j$ in $1 \ldots W$) **do**

    $Y_{i,j} \leftarrow \sum_k w_{i,j,k} x_k$ {Perform the convolution}

  **end for**

---

# 5 Experimental Results

We performed experiments on two sets of images: MNIST digits dataset and NORB object classi-fication dataset [1]. We used the "jittered objects and cluttered background" image set from NORB. The MNIST images are particular in that they have a white background, whereas the NORB images have more varying backgrounds. The NORB images are originally of dimension $108 \times 108$; we subsampled them by $4 \times 4$ averaging into $27 \times 27$ images. The experiments have been performed with $k = 4$ neighbors for the Isomap embedding. Smaller values of $k$ often led to unconnected neighborhood graphs, which Isomap cannot deal with.

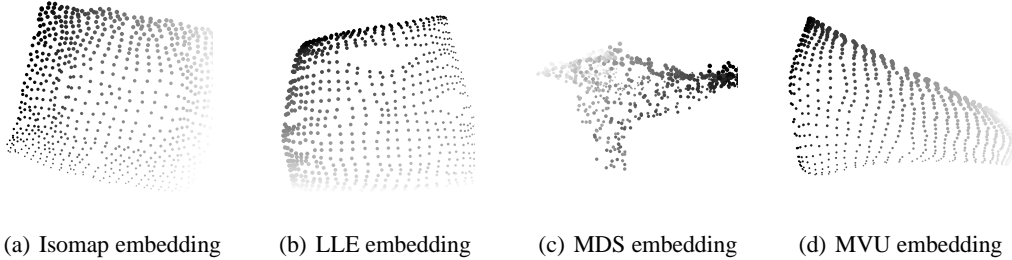

(a) Isomap embedding     (b) LLE embedding     (c) MDS embedding     (d) MVU embedding

Figure 1: Examples of embeddings discovered by Isomap, LLE, MDS and MVU with 250 training images from NORB. Each of the original pixel is placed at the location discovered by the algorithm. Size of the circle and gray level indicate the original true location of the pixel. Manifold learning produces coordinates with an arbitrary rotation. Isomap appears most robust, and MDS the worst method, for this task.

In Figure 1 we compare four different manifold learning algorithms on the NORB images: Isomap, LLE, MDS and MVU. Figure 2 explains why Isomap is giving good results, especially in comparison with MDS. One the one hand, MDS is using the pseudo-distance defined in equation 1, whose relationship with the real distance between two pixels in the original image is linear only in a small neighborhood. On the other hand, Isomap uses the geodesic distances in the neighborhood graph, whose relationship with the real distance is really close to linear.

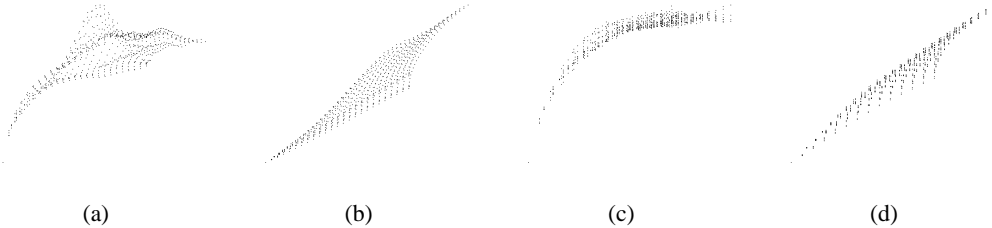

(a)      (b)      (c)      (d)

Figure 2: (a) and (c): Pseudo-distance $D_{ij}$ (using formula 1) vs. the true distance on the grid.
(b) and (d): Geodesic distance in neighborhood graph vs. the true distance on the grid.
The true distance is on the horizontal axis for all figures.
(a) and (b) are for a point in the upper-left corner, (c) and (d) for a point in the center.

Figure 3 shows the embeddings obtained on the NORB data using different numbers of examples. In order to quantitatively evaluate the reconstruction, we applied on each embedding the similarity transformation that minimizes the Root of the Mean Squared Error (RMSE) between the coordinates of each pixel on the embedding, and their coordinates on the original grid, before measuring the residual error. This minimization is justified because the discovered embedding could be arbitrarily rotated, isotropically scaled, and mirrored. 100 examples are enough to get a reasonable embedding, and with 2000 or more a very good embedding is obtained: the RMSE for 2000 examples is $1.13$, meaning that in expectation, each pixel is off by slightly more than one.

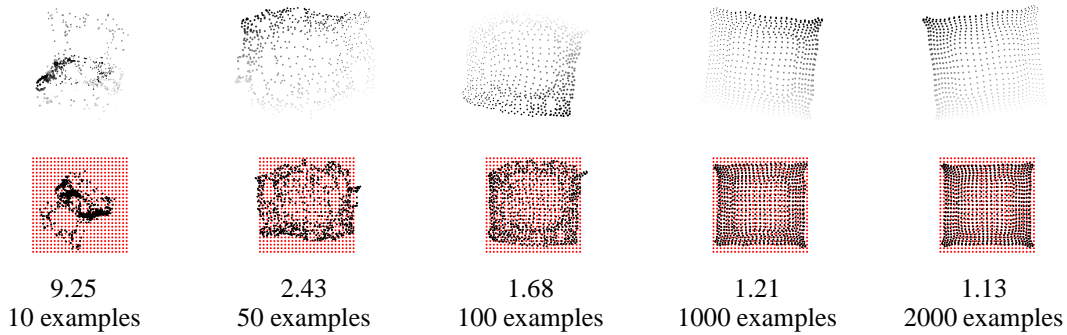

| 9.25 | 2.43 | 1.68 | 1.21 | 1.13 |
|------|------|------|------|------|
| 10 examples | 50 examples | 100 examples | 1000 examples | 2000 examples |

Figure 3: Embedding discovered by Isomap on the NORB dataset, with different numbers of training samples (top row). Second row shows the same embeddings aligned (by a similarity transformation) on the original grid, third row shows the residual error (RMSE) after the alignment.

Figure 4 shows the whole process of transforming an original image (with pixels possibly permuted) into an embedded image and finally into a reconstructed image as per algorithms 1 and 2.

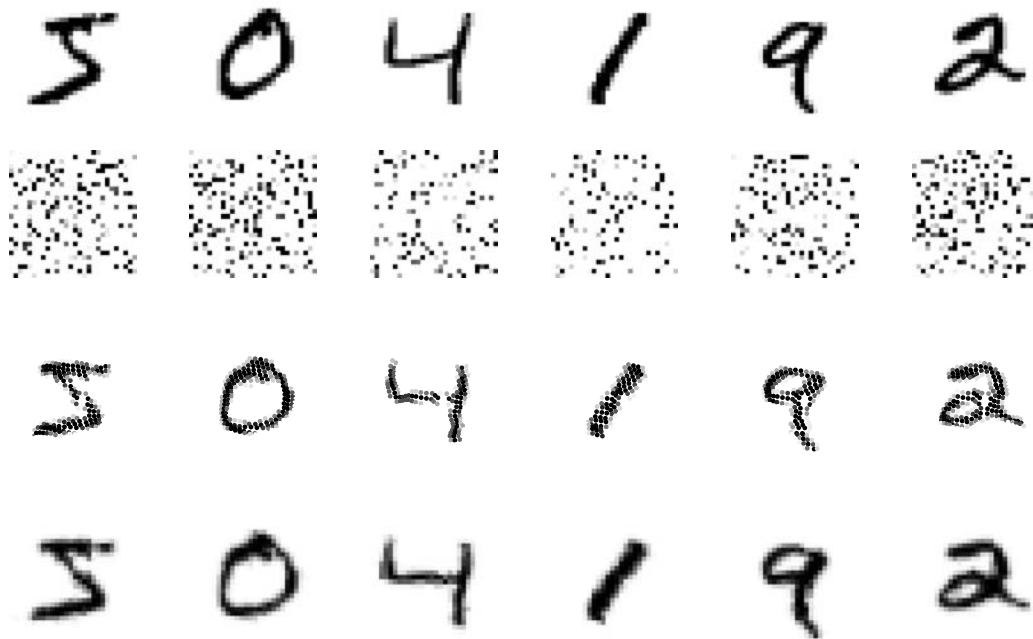

Figure 4: Example of the process of transforming an MNIST image (top) from which pixel order is unknown (second row) into its embedding (third row) and finally reconstructed as an image after rotation and convolution (bottom). In the third row, we show the intensity associated to each original pixel by the grey level in a circle located at the pixel coordinates discovered by Isomap.

We also performed experiments with acoustic spectral data to see if the time-frequency topology can be recovered. The acoustic data come from the first 100 blues pieces of a publically available genre classification dataset [14]. The FFT is computed for each frame and there are 86 frames per second. The first 30 frequency bands are kept, each covering 21.51 Hz. We used examples formed by 30-frame spectrograms, i.e., just like images of size $30 \times 30$. Using the first 600,000 audio samples from each recording yielded 2600 30-frames images, on which we applied our technique. Figure 5 shows the resulting embedding when we removed the 30 coordinates of lowest standard deviation ($\delta = .15$).

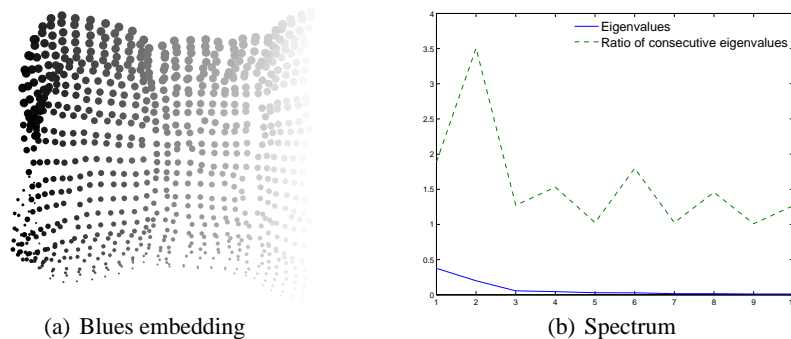

(a) Blues embedding                    (b) Spectrum

Figure 5: Embedding and spectrum decay for sequences of blues music.

## 6    Discussion

Although [8] argue that learning the right permutation of pixels with a flat prior might be too difficult (either in a lifetime or through evolution), our results suggest otherwise.

How do we interpret that apparent contradiction?

The main element of explanation that we see is that the space of permutations of $d$ numbers is not such a large class of functions. There are approximately $N = \sqrt{2\pi d}\left(\frac{d}{e}\right)^d$ permutations (Stirling approximation) of $d$ numbers. Since this is a finite class of functions, its VC-dimension [15] is

$$h = \log N \approx d \log d - d.$$

Hence if we had a bounded criterion (say taking values in $[0, 1]$) to compare different permutations and we used $n$ examples (i.e., $n$ images, here), we would expect the difference between generalization error and test error to be bounded [15] by $\frac{1}{2}\sqrt{\frac{2\log N/\eta}{n}}$ with probability $1-\eta$. Hence, with $n$ a multiple of $d \log d$, we would expect that one could approximately learn a good permutation. When $d = 400$ (the number of pixels with non-negligible variance in MNIST images), $d \log d - d \approx 2000$. This is more than what we have found necessary to recover a "good" representation of the images, but on the other hand there are equivalent classes within the set of permutations that give as good results as far as our objective and subjective criteria are concerned: we do not care about image symmetries, rotations, and small errors in pixel placement.

What is the selection criterion that we have used to recover the image structure? Mainly we have used an additional prior which gives a preference to an order for which nearby pixels have similar distributions. How specific to natural images and how strong is that prior? This may be an application of a more general principle that could be advantageous to learning algorithms as well as to brains. When we are trying to compute useful functions from raw data, it is important to discover dependencies between the input random variables. If we are going to perform computations on subsets of variables at a time (which would seem necessary when the number of inputs is very large, to reduce the amount of connecting hardware), it would seem wiser that these computations combine variables that have dependencies with each other. That directly gives rise to the notion of local connectivity between neurons associated to nearby spatial locations, in the case of brains, the same notion that is exploited in convolutional neural networks.

The fact that nearby pixels are more correlated is true at many scales in natural images. This is well known and explains why Gabor-like filters often emerge when trying to learn good filters for images, e.g., by ICA [9] or Products of Experts [3, 11].

In addition to the above arguments, there is another important consideration to keep in mind. The way in which we score permutations is not the way that one would score functions in an ordinary learning experiment. Indeed, by using the distributional similarity between pairs of pixels, we get not just a scalar score but $d(d-1)/2$ scores. Since our "scoring function" is much more informative, it is not surprising that it allows us to generalize from many fewer examples.

# 7 Conclusion and Future Work

We proved here that, even with a small number of examples, we are able to recover almost perfectly the 2-D topology of images. This allows us to use image-specific learning algorithms without specifying any prior other than the dimensionnality of the coordinates. We also showed that this algorithm performed well on sound data, even though the topology might be less obvious in that case.

However, in this paper, we only considered the simple case where we knew in advance the dimensionnality of the coordinates. One could easily apply this algorithm to data whose intrinsic dimensionality of the coordinates is unknown. In that case, one would not convert the embedding to a grid image but rather keep it and connect only the inputs associated to close coordinates (performing a $k$ nearest neighbor for instance). It is not known if such an embedding might be useful for other types of data than the ones discussed above.

### Acknowledgements

The authors would like to thank James Bergstra for helping with the audio data. They also want to acknowledge the support from several funding agencies: NSERC, the Canada Research Chairs, and the MITACS network.

## Footnotes

[1] Both can be obtained from Yann Le Cun's web site: `http://yann.lecun.com/`.

# References

[1] S. Abdallah and M. Plumbley. Geometry dependency analysis. Technical Report C4DM-TR06-05, Center for Digital Music, Queen Mary, University of London, 2006.

[2] Y. Bengio and Y. Le Cun. Scaling learning algorithms towards AI. In L. Bottou, O. Chapelle, D. DeCoste, and J. Weston, editors, *Large Scale Kernel Machines*. MIT Press, 2007.

[3] G. Hinton, M. Welling, Y. Teh, and S. Osindero. A new view of ica. In *Proceedings of ICA-2001*, San Diego, CA, 2001.

[4] A. Hyvärinen, P. O. Hoyer, and M. Inki. Topographic independent component analysis. *Neural Computation*, 13(7):1527–1558, 2001.

[5] K. J. Lang and G. E. Hinton. The development of the time-delay neural network architecture for speech recognition. Technical Report CMU-CS-88-152, Carnegie-Mellon University, 1988.

[6] Y. LeCun, B. Boser, J. Denker, D. Henderson, R. Howard, W. Hubbard, and L. Jackel. Backpropagation applied to handwritten zip code recognition. *Neural Computation*, 1(4):541–551, 1989.

[7] Y. LeCun, L. Bottou, Y. Bengio, and P. Haffner. Gradient based learning applied to document recognition. *Proceedings of the IEEE*, 86(11):2278–2324, November 1998.

[8] Y. LeCun and J. S. Denker. Natural versus universal probability complexity, and entropy. In *IEEE Workshop on the Physics of Computation*, pages 122–127. IEEE, 1992.

[9] T.-W. Lee and M. S. Lewicki. Unsupervised classification segmentation and enhancement of images using ica mixture models. *IEEE Trans. Image Proc.*, 11(3):270–279, 2002.

[10] D. Lowe. Distinctive image features from scale-invariant keypoints. *International Journal of Computer Vision*, 60(2):91–110, 2004.

[11] S. Osindero, M. Welling, and G. Hinton. Topographic product models applied to natural scene statistics. *Neural Computation*, 18:381–344, 2005.

[12] S. Roweis and L. Saul. Nonlinear dimensionality reduction by locally linear embedding. *Science*, 290(5500):2323–2326, Dec. 2000.

[13] J. Tenenbaum, V. de Silva, and J. Langford. A global geometric framework for nonlinear dimensionality reduction. *Science*, 290(5500):2319–2323, Dec. 2000.

[14] G. Tzanetakis and P. Cook. Musical genre classification of audio signals. *IEEE Transactions on Speech and Audio Processing*, 10(5):293–302, Jul 2002.

[15] V. Vapnik. *Estimation of Dependences Based on Empirical Data*. Springer-Verlag, Berlin, 1982.

[16] A. Waibel. Modular construction of time-delay neural networks for speech recognition. *Neural Computation*, 1:39–46, 1989.

[17] K. Q. Weinberger and L. K. Saul. An introduction to nonlinear dimensionality reduction by maximum variance unfolding. In *Proceedings of the National Conference on Artificial Intelligence (AAAI)*, Boston, MA, 2006.

